# An Efficient, Exact Algorithm for Solving Tree-Structured Graphical Games

**Michael L. Littman**
AT&T Labs–Research
Florham Park, NJ 07932-0971
mlittman@research.att.com

**Michael Kearns**
Department of Computer & Information Science
University of Pennsylvania
Philadelphia, PA 19104-6389
mkearns@cis.upenn.edu

**Satinder Singh**
Syntek Capital
New York, NY 10019-4460
baveja@cs.colorado.edu

## Abstract

We describe a new algorithm for computing a Nash equilibrium in *graphical games*, a compact representation for multi-agent systems that we introduced in previous work. The algorithm is the first to compute equilibria both efficiently and exactly for a non-trivial class of graphical games.

## 1   Introduction

Seeking to replicate the representational and computational benefits that graphical models have provided to probabilistic inference, several recent works have introduced graph-theoretic frameworks for the study of multi-agent systems (La Mura 2000; Koller and Milch 2001; Kearns et al. 2001). In the simplest of these formalisms, each vertex represents a single agent, and the edges represent pairwise interaction between agents. As with many familiar network models, the macroscopic behavior of a large system is thus implicitly described by its local interactions, and the computational challenge is to extract the global states of interest. Classical game theory is typically used to model multi-agent interactions, and the global states of interest are thus the so-called Nash equilibria, in which no agent has a unilateral incentive to deviate.

In a recent paper (Kearns et al. 2001), we introduced such a graphical formalism for multi-agent game theory, and provided two algorithms for computing Nash equilibria when the underlying graph is a tree (or is sufficiently sparse). The first algorithm

computes approximations to all Nash equilibria, in time polynomial in the size of the representation and the quality of the desired approximation. A second and related algorithm computes all Nash equilibria exactly, but in time exponential in the number of agents. We thus left open the problem of efficiently computing exact equilibria in sparse graphs.

In this paper, we describe a new algorithm that solves this problem. Given as input a graphical game that is a tree, the algorithm computes in polynomial time an exact Nash equilibrium for the global multi-agent system. The main advances involve the definition of a new data structure for representing "upstream" or partial Nash equilibria, and a proof that this data structure can always be extended to a global equilibrium. The new algorithm can also be extended to efficiently accommodate parametric representations of the local game matrices, which are analogous to parametric conditional probability tables (such as noisy-OR and sigmoids) in Bayesian networks.

The analogy between graphical models for multi-agent systems and probabilistic inference is tempting and useful to an extent. The problem of computing Nash equilibria in a graphical game, however, appears to be considerably more difficult than computing conditional probabilities in Bayesian networks. Nevertheless, the analogy and the work presented here suggest a number of interesting avenues for further work in the intersection of game theory, network models, probabilistic inference, statistical physics, and other fields.

The paper is organized as follows. Section 2 introduces graphical games and other necessary notation and definitions. Section 3 presents our algorithm and its analysis, and Section 4 gives a brief conclusion.

## 2 Preliminaries

An $n$-player, two-action [1] game is defined by a set of $n$ matrices $M_i$ ($1 \leq i \leq n$), each with $n$ indices. The entry $M_i(x_1, \ldots, x_n) = M_i(\vec{x})$ specifies the payoff to player $i$ when the joint action of the $n$ players is $\vec{x} \in \{0, 1\}^n$. Thus, each $M_i$ has $2^n$ entries. If a game is given by simply listing the $2^n$ entries of each of the $n$ matrices, we will say that it is represented in *tabular* form.

The actions 0 and 1 are the *pure strategies* of each player, while a *mixed* strategy for player $i$ is given by the probability $p_i \in [0, 1]$ that the player will play 1. For any joint mixed strategy, given by a product distribution $\vec{p}$, we define the expected payoff to player $i$ as $M_i(\vec{p}) = \mathbf{E}_{\vec{x} \sim \vec{p}}[M_i(\vec{x})]$, where $\vec{x} \sim \vec{p}$ indicates that each $x_j$ is 1 with probability $p_j$ and 0 with probability $1 - p_j$.

We use $\vec{p}[i : p_i']$ to denote the vector that is the same as $\vec{p}$ except in the $i$th component, where the value has been changed to $p_i'$. A *Nash equilibrium* for the game is a mixed strategy $\vec{p}$ such that for any player $i$, and for any value $p_i' \in [0, 1]$, $M_i(\vec{p}) \geq M_i(\vec{p}[i : p_i'])$. (We say that $p_i$ is a *best response* to $\vec{p}$.) In other words, no player can improve its expected payoff by deviating unilaterally from a Nash equilibrium. The classic theorem of Nash (1951) states that for any game, there exists a Nash equilibrium in the space of joint mixed strategies (product distributions).

An $n$-player *graphical game* is a pair $(G, \mathcal{M})$, where $G$ is an undirected graph [2] on $n$

vertices and $\mathcal{M}$ is a set of $n$ matrices $M_i$ $(1 \leq i \leq n)$, called the *local game matrices*. Player $i$ is represented by a vertex labeled $i$ in $G$. We use $N_G(i) \subseteq \{1, \ldots, n\}$ to denote the set of *neighbors* of player $i$ in $G$—those vertices $j$ such that the undirected edge $(i, j)$ appears in $G$. By convention, $N_G(i)$ always includes $i$ itself. The interpretation is that each player is in a game with only his neighbors in $G$. Thus, if $|N_G(i)| = k$, the matrix $M_i$ has $k$ indices, one for each player in $N_G(i)$, and if $\vec{x} \in [0, 1]^k$, $M_i(\vec{x})$ denotes the payoff to $i$ when his $k$ neighbors (which include himself) play $\vec{x}$. The expected payoff under a mixed strategy $\vec{p} \in [0, 1]^k$ is defined analogously. Note that in the two-action case, $M_i$ has $2^k$ entries, which may be considerably smaller than $2^n$.

Since we identify players with vertices in $G$, it will be easier to treat vertices symbolically (such as $U, V$ and $W$) rather than by integer indices. We thus use $M_V$ to denote the local game matrix for the player identified with vertex $V$.

Note that our definitions are entirely representational, and alter nothing about the underlying game theory. Thus, every graphical game has a Nash equilibrium. Furthermore, every game can be trivially represented as a graphical game by choosing $G$ to be the complete graph and letting the local game matrices be the original tabular form matrices. Indeed, in some cases, this may be the most compact graphical representation of the tabular game. However, exactly as for Bayesian networks and other graphical models for probabilistic inference, any game in which the local neighborhoods in $G$ can be bounded by $k \ll n$, exponential *space* savings accrue. The algorithm presented here demonstrates that for trees, exponential *computational* benefits may also be realized.

## 3   The Algorithm

If $(G, \mathcal{M})$ is a graphical game in which $G$ is a tree, then we can always designate some vertex $Z$ as the root. For any vertex $V$, the single neighbor of $V$ on the path from $V$ to $Z$ shall be called the child of $V$, and the (possibly many) neighbors of $V$ on paths towards the leaves shall be called the parents of $V$. Our algorithm consists of two passes: a *downstream* pass in which local data structures are passed from the leaves towards the root, and an *upstream* pass progressing from the root towards the leaves.

Throughout the ensuing discussion, we consider a fixed vertex $V$ with parents $U_1, \ldots, U_k$ and child $W$. On the downstream pass of our algorithm, vertex $V$ will compute and pass to its child $W$ a *breakpoint policy*, which we now define.

**Definition 1** *A breakpoint policy for $V$ consists of an ordered set of $W$-breakpoints $w_0 = 0 < w_1 < w_2 < \cdots < w_{t-1} < w_t = 1$ and an associated set of $V$-values $v_1, \ldots, v_t$. The interpretation is that for any $w \in [0, 1]$, if $w_{i-1} < w < w_i$ for some index $i$ and $W$ plays $w$, then $V$ shall play $v_i$; and if $w = w_i$ for some index $i$, then $V$ shall play any value between $v_i$ and $v_{i+1}$. We say such a breakpoint policy has $t - 1$ breakpoints.*

A breakpoint policy for $V$ can thus be seen as assigning a value (or range of values) to the mixed strategy played by $V$ in response to the play of its child $W$. In a slight abuse of notation, we will denote this breakpoint policy as a *function $F_V(w)$*, with the understanding that the assignment $V = F_V(w)$ means that $V$ plays either the fixed value determined by the breakpoint policy (in the case that $w$ falls between breakpoints), or plays any value in the interval determined by the breakpoint policy (in the case that $w$ equals some breakpoint).

Let $G^V$ denote the subtree of $G$ with root $V$, and let $\mathcal{M}^V_{W=w}$ denote the subset of the set of local game matrices $\mathcal{M}$ corresponding to the vertices in $G^V$, except that the matrix $M_V$ is collapsed one index by setting $W = w$, thus marginalizing $W$ out. On its downstream pass, our algorithm shall maintain the invariant that if we set the child $W = w$, then there is a Nash equilibrium for the graphical game $(G^V, \mathcal{M}^V_{W=w})$ (an *upstream Nash*) in which $V = F_V(w)$. If this property is satisfied by $F_V(w)$, we shall say that $F_V(w)$ is a *Nash* breakpoint policy for $V$. Note that since $(G^V, \mathcal{M}^V_{W=w})$ is just another graphical game, it of course has (perhaps many) Nash equilibria, and $V$ is assigned some value in each. The trick is to commit to one of these values (as specified by $F_V(w)$) that can be *extended* to a Nash equilibrium for the entire tree $G$, before we have even processed the tree below $V$. Accomplishing this efficiently and exactly is one of the main advances in this work over our previous algorithm (Kearns et al. 2001).

The algorithm and analysis are inductive: $V$ computes a Nash breakpoint policy $F_V(w)$ from Nash breakpoint policies $F_{U_1}(v), \ldots, F_{U_k}(v)$ passed down from its parents (and from the local game matrix $M_V$). The complexity analysis bounds the number of breakpoints for any vertex in the tree. We now describe the inductive step and its analysis.

### 3.1 Downstream Pass

For any setting $\vec{u} \in [0,1]^k$ for $\vec{U}$ and $w \in [0,1]$ for $W$, let us define

$$\Delta_V(\vec{u}, w) \equiv M_V(1, \vec{u}, w) - M_V(0, \vec{u}, w).$$

The sign of $\Delta_V(\vec{u}, w)$ tells us $V$'s best response to the setting of the local neighborhood $\vec{U} = \vec{u}, W = w$; positive sign means $V = 1$ is the best response, negative that $V = 0$ is the best response, and 0 that $V$ is indifferent and may play any mixed strategy. Note also that we can express $\Delta_V(\vec{u}, w)$ as a linear function of $w$:

$$\Delta_V(\vec{u}, w) = \Delta_V(\vec{u}, 0) + w(\Delta_V(\vec{u}, 1) - \Delta_V(\vec{u}, 0)).$$

For the base case, suppose $V$ is a leaf with child $W$; we want to describe the Nash breakpoint policy for $V$. If for all $w \in [0,1]$, the function $\Delta_V(w)$ is non-negative (non-positive, respectively), $V$ can choose 1 (0, respectively) as a best response (which in this base case is an upstream Nash) to all values $W = w$. Otherwise, $\Delta_V(w)$ crosses the $w$-axis, separating the values of $w$ for which $V$ should choose 1, 0, or be indifferent (at the crossing point). Thus, this crossing point becomes the single breakpoint in $F_V(w)$. Note that if $V$ is indifferent for all values of $w$, we assume without loss of generality that $V$ plays 1.

The following theorem is the centerpiece of the analysis.

**Theorem 2** *Let vertex $V$ have parents $U_1, \ldots, U_k$ and child $W$, and assume $V$ has received Nash breakpoint policies $F_{U_i}(v)$ from each parent $U_i$. Then $V$ can efficiently compute a Nash breakpoint policy $F_V(w)$. The number of breakpoints is no more than two plus the total number of breakpoints in the $F_{U_i}(v)$ policies.*

**Proof:** Recall that for any fixed value of $v$, the breakpoint policy $F_{U_i}(v)$ specifies either a specific value for $U_i$ (if $v$ falls between two breakpoints of $F_{U_i}(v)$), or a range of allowed values for $U_i$ (if $v$ is equal to a breakpoint). Let us assume without loss of generality that no two $F_{U_i}(v)$ share a breakpoint, and let $v_0 = 0 < v_1 < \cdots < v_s = 1$ be the ordered union of the breakpoints of the $F_{U_i}(v)$. Thus for any breakpoint $v_\ell$, there is at most one distinguished parent $U_j$ (that we shall call the *free* parent) for which $F_{U_j}(v_\ell)$ specifies an allowed interval of play for $U_j$. All other $U_i$ are assigned

fixed values by $F_{U_i}(v_\ell)$. For each breakpoint $v_\ell$, we now define the set of values for the child $W$ that, as we let the free parent range across its allowed interval, permit $V$ to play any mixed strategy as a best response.

**Definition 3** *Let $v_0 = 0 < v_1 < \cdots < v_s = 1$ be the ordered union of the breakpoints of the parent policies $F_{U_i}(v)$. Fix any breakpoint $v_\ell$, and assume without loss of generality that $U_1$ is the free parent of $V$ for $V = v_\ell$. Let $[a, b]$ be the allowed interval of $U_1$ specified by $F_{U_1}(v_\ell)$, and let $u_i = F_{U_i}(v_\ell)$ for all $2 \leq i \leq k$. We define*

$$\mathcal{W}_\ell = \{w \in [0, 1] : (\exists u_1 \in [a, b]) \Delta_V(u_1, u_2, \ldots, u_k, w) = 0\}.$$

*In words, $\mathcal{W}_\ell$ is the set of values that $W$ can play that allow $V$ to play any mixed strategy, preserving the existence of an upstream Nash from $V$ given $W = w$.*

The next lemma, which we state without proof and is a special case of Lemma 6 in Kearns et al. (2001), limits the complexity of the sets $\mathcal{W}_\ell$. It also follows from the earlier work that $\mathcal{W}_\ell$ can be computed in time proportional to the size of $V$'s local game matrix — $O(2^k)$ for a vertex with $k$ parents.

We say that an interval $[a, b] \subseteq [0, 1]$ is *floating* if both $a \neq 0$ and $b \neq 1$.

**Lemma 4** *For any breakpoint $v_\ell$, the set $\mathcal{W}_\ell$ is either empty, a single interval, or the union of two intervals that are not floating.*

We wish to create the (inductive) Nash breakpoint policy $F_V(w)$ from the sets $\mathcal{W}_\ell$ and the $F_{U_i}$ policies. The idea is that if $w \in \mathcal{W}_\ell$ for some breakpoint index $\ell$, then by definition of $\mathcal{W}_\ell$, if $W$ plays $w$ and the $U_i$s play according to the setting determined by the $F_{U_i}$ policies (including a fixed setting for the free parent of $V$), *any* play by $V$ is a best response—so in particular, $V$ may play the breakpoint value $v_\ell$, and thus extend the Nash solution constructed, as the $U_i$s can also all be best responses. For $b \in \{0, 1\}$, we define $\mathcal{W}^b$ as the set of values $w$ such that if $W = w$ and the $U_i$s are set according to their breakpoint policies for $V = b$, $V = b$ is a best response. To create $F_V(w)$ as a total function, we must first show that every $w \in [0, 1]$ is contained in some $\mathcal{W}_\ell$ or $\mathcal{W}^0$ or $\mathcal{W}^1$.

**Lemma 5** *Let $v_0 = 0 < v_1 < \cdots < v_s = 1$ be the ordered union of the breakpoints of the $F_{U_i}(v)$ policies. Then for any value $w \in [0, 1]$, either $w \in \mathcal{W}^b$ for some $b \in \{0, 1\}$, or there exists an index $\ell$ such that $w \in \mathcal{W}_\ell$.*

**Proof:** Consider any fixed value of $w$, and for each open interval $(v_j, v_{j+1})$ determined by adjacent breakpoints, label this interval by $V$'s best response (0 or 1) to $W = w$ and $\vec{U}$ set according to the $F_{U_i}$ policies for this interval. If either the leftmost interval $[0, v_1]$ is labeled with 0 or the rightmost interval $[v_{s-1}, 1]$ is labeled with 1, then $w$ is included in $\mathcal{W}^0$ or $\mathcal{W}^1$, respectively ($V$ playing 0 or 1 is a best response to what the $U_i$s will play in response to a 0 or 1). Otherwise, since the labeling starts at 1 on the left and ends at 0 on the right, there must be a breakpoint $v_\ell$ such that $V$'s best response changes over this breakpoint. Let $U_i$ be the free parent for this breakpoint. By continuity, there must be a value of $U_i$ in its allowed interval for which $V$ is indifferent to playing 0 or 1, so $w \in \mathcal{W}_\ell$. This completes the proof of Lemma 5.

Armed with Lemmas 4 and 5, we can now describe the construction of $F_V(w)$. Since every $w$ is contained in some $\mathcal{W}_\ell$ (Lemma 5), and since every $\mathcal{W}_\ell$ is the union of at most two intervals (Lemma 4), we can uniquely identify the set $\mathcal{W}_{\ell_1}$ that covers the largest (leftmost) interval containing $w = 0$; let $[0, a]$ be this interval. Continuing in the same manner to the right, we can identify the unique set $\mathcal{W}_{\ell_2}$ that contains

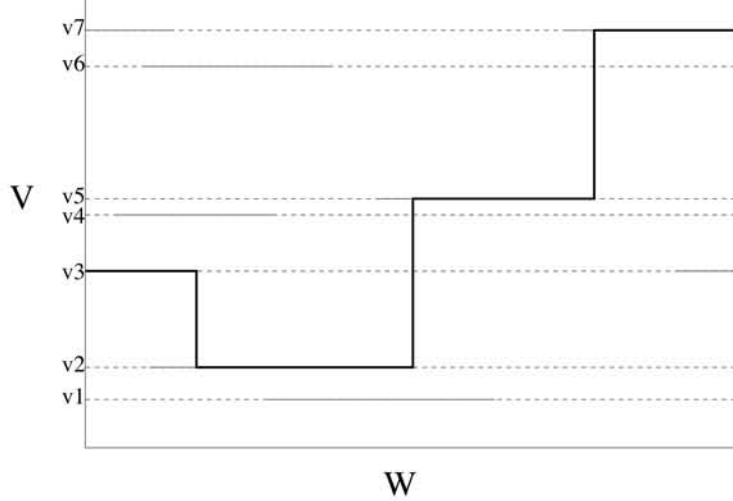

Figure 1: Example of the inductive construction of $F_V(w)$. The dashed horizontal lines show the $v_\ell$-breakpoints determined by the parent policies $F_{U_i}(v)$. The solid intervals along these breakpoints are the sets $\mathcal{W}_\ell$. As shown in Lemma 4, each of these sets consists of either a single (possibly floating) interval, or two non-floating intervals. As shown in Lemma 5, each value of $w$ is covered by some $\mathcal{W}_\ell$. The construction of $F_V(w)$ (represented by a thick line) begins on the left, and always next "jumps" to the interval allowing greatest progress to the right.

$w = a$ and extends farthest to the right of $a$. Any overlap between $\mathcal{W}_{\ell_1}$ and $\mathcal{W}_{\ell_2}$ can be arbitrarily assigned coverage by $\mathcal{W}_{\ell_1}$, and $\mathcal{W}_{\ell_2}$ "trimmed" accordingly; see Figure 1. This process results in a Nash breakpoint policy $F_V(w)$.

Finally, we bound the number of breakpoints in the $F_V(w)$ policy. By construction, each of its breakpoints must be the rightmost portion of some interval in $\mathcal{W}^0$, $\mathcal{W}^1$, or some $\mathcal{W}_\ell$. After the first breakpoint, each of these sets contributes at most one new breakpoint (Lemma 4). The final breakpoint is at $w = 1$ and does not contribute to the count (Definition 1). There is at most one $\mathcal{W}_\ell$ for each breakpoint in each $F_{U_i}(v)$ policy, plus $\mathcal{W}^0$ and $\mathcal{W}^1$, plus the initial leftmost interval and minus the final breakpoint, so the total breakpoints in $F_V(w)$ can be no more than two plus the total number of breakpoints in the $F_{U_i}(v)$ policies. Therefore, the root of a size $n$ tree will have a Nash breakpoint policy with no more than $2n$ breakpoints.

This completes the proof of Theorem 2.

## 3.2 Upstream Pass

The downstream pass completes when each vertex in the tree has had its Nash breakpoint policy computed. For simplicity of description, imagine that the root of the tree includes a dummy child with constant payoffs and no influence on the root, so the root's breakpoint policy has the same form as the others in the tree.

To produce a Nash equilibrium, our algorithm performs an upstream pass over the tree, starting from the root. Each vertex is told by its child what value to play, as well as the value the child itself will play. The algorithm ensures that all downstream vertices are Nash (playing best response to their neighbors). Given this information, each vertex computes a value for each of its parents so that its

own assigned action is a best response. This process can be initiated by the dummy vertex picking an arbitrary value for itself, and selecting the root's value according to its Nash breakpoint policy.

Inductively, we have a vertex $V$ connected to parents $U_1, \ldots, U_k$ (or no parents if $V$ is a leaf) and child $W$. The child of $V$ has informed $V$ to chose $V = v$ and that it will play $W = w$. To decide on values for $V$'s parents to enforce $V$ playing a best response, we can look at the Nash breakpoint policies $F_{U_i}(v)$, which provide a value (or range of values) for $U_i$ as a function of $v$ that guarantee an upstream Nash. The value $v$ can be a breakpoint for at most one $U_i$. For each $U_i$, if $v$ is not a breakpoint in $F_{U_i}(v)$, then $U_i$ should be told to select $U_i = F_{U_i}(v)$. If $v$ is a breakpoint in $F_{U_i}(v)$, then $U_i$'s value can be computed by solving $\Delta_V(u_1, \ldots, u_i, \ldots, u_k, w) = 0$; this is the value of $u_i$ that makes $V$ indifferent. The equation is linear in $u_i$ and has a solution by the construction of the Nash breakpoint policies on the downstream pass. Parents are passed their assigned values as well as the fact that $V = v$.

When the upstream pass completes, each vertex has a concrete choice of action such that jointly they have formed a Nash equilibrium.

The total running time of the algorithm can be bounded as follows. Each vertex is involved in a computation in the downstream pass and in the upstream pass. Let $t$ be the total number of breakpoints in the breakpoint policy for a vertex $V$ with $k$ parents. Sorting the breakpoints and computing the $\mathcal{W}_\ell$ sets and computing the new breakpoint policy can be completed in $O(t \log t + t 2^k)$. In the upstream pass, only one breakpoint is considered, so $O(\log t + 2^k)$ is sufficient for passing breakpoints to the parents. By Theorem 2, $t \leq 2n$, so the entire algorithm executes in time $O(n^2 \log n + n^2 2^k)$, where $k$ is the largest number of neighbors of any vertex in the network.

The algorithm can be implemented to take advantage of local game matrices provided in a parameterized form. For example, if each vertex's payoff is solely a function of the number of 1s played by the vertex's neighbors, the algorithm takes $O(n^2 \log n + n^2 k)$, eliminating the exponential dependence on $k$.

# 4 Conclusion

The algorithm presented in this paper finds a single Nash equilibrium for a game represented by a tree-structured network. By building representations of *all* equilibria, our earlier algorithm (Kearns et al. 2001) was able to select equilibria efficiently according to criteria like maximizing the total expected payoff for all players. The polynomial-time algorithm described in this paper throws out potential equilibria at many stages, most significantly during the construction of the Nash breakpoint policies. An interesting area for future work is to manipulate this process to produce equilibria with particular properties.

## Footnotes

[1] At present, no polynomial-time algorithm is known for finding Nash equilibria even in 2-player games with more than two actions, so we leave the extension of our work to the multi-action setting for future work.

[2] The directed tree-structured case is trivial and is not addressed in this paper.

# References

Michael Kearns, Michael L. Littman, and Satinder Singh. Graphical models for game theory. In *Proceedings of the 17th Conference on Uncertainty in Artificial Intelligence (UAI)*, pages 253–260, 2001.

Daphne Koller and Brian Milch. Multi-agent influence diagrams for representing and solving games. Submitted, 2001.

Pierfrancesco La Mura. Game networks. In *Proceedings of the 16th Conference on Uncertainty in Artificial Intelligence (UAI)*, pages 335–342, 2000.

J. F. Nash. Non-cooperative games. *Annals of Mathematics*, 54:286–295, 1951.
